# Learning to Align from Scratch

**Gary B. Huang**[1]    **Marwan A. Mattar**[1]    **Honglak Lee**[2]    **Erik Learned-Miller**[1]
[1]University of Massachusetts, Amherst, MA
{gbhuang,mmattar,elm}@cs.umass.edu
[2]University of Michigan, Ann Arbor, MI
honglak@eecs.umich.edu

## Abstract

Unsupervised joint alignment of images has been demonstrated to improve performance on recognition tasks such as face verification. Such alignment reduces undesired variability due to factors such as pose, while only requiring weak supervision in the form of poorly aligned examples. However, prior work on unsupervised alignment of complex, real-world images has required the careful selection of feature representation based on hand-crafted image descriptors, in order to achieve an appropriate, smooth optimization landscape. In this paper, we instead propose a novel combination of unsupervised joint alignment with unsupervised feature learning. Specifically, we incorporate deep learning into the *congealing* alignment framework. Through deep learning, we obtain features that can represent the image at differing resolutions based on network depth, and that are tuned to the statistics of the specific data being aligned. In addition, we modify the learning algorithm for the restricted Boltzmann machine by incorporating a group sparsity penalty, leading to a topographic organization of the learned filters and improving subsequent alignment results. We apply our method to the Labeled Faces in the Wild database (LFW). Using the aligned images produced by our proposed unsupervised algorithm, we achieve higher accuracy in face verification compared to prior work in both unsupervised and supervised alignment. We also match the accuracy for the best available commercial method.

## 1   Introduction

One of the most challenging aspects of image recognition is the large amount of intra-class variability, due to factors such as lighting, background, pose, and perspective transformation. For tasks involving a specific object category, such as face verification, this intra-class variability can often be much larger than inter-class differences. This variability can be seen in Figure 1, which shows sample images from Labeled Faces in the Wild (LFW), a data set used for benchmarking unconstrained face verification performance. The task in LFW is, given a pair of face images, determine if both faces are of the same person (matched pair), or if each shows a different person (mismatched pair).

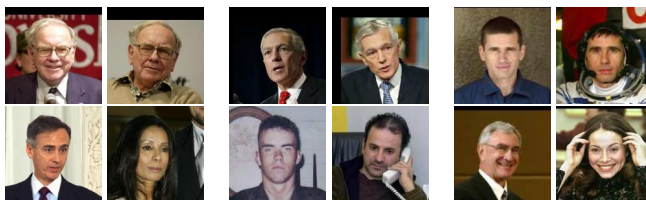

Figure 1: Sample images from LFW: matched pairs (top row) and mismatched pairs (bottom row)

Recognition performance can be significantly improved by removing undesired intra-class variability, by first aligning the images to some canonical pose or configuration. For instance, face verification accuracy can be dramatically increased through image alignment, by detecting facial feature points on the image and then warping these points to a canonical configuration. This alignment process can lead to significant gains in recognition accuracy on real-world face verification, even

for algorithms that were explicitly designed to be robust to some misalignment [1]. Therefore, the majority of face recognition systems evaluated on LFW currently make use of a preprocessed version of the data set known as LFW-a,[1] where the images have been aligned by a commercial fiducial point-based supervised alignment method [2]. Fiducial point (or landmark-based) alignment algorithms [1, 3–5], however, require a large amount of supervision or manual effort. One must decide which fiducial points to use for the specific object class, and then obtain many example image patches of these points. These methods are thus hard to apply to new object classes, since all of this manual collection of data must be re-done, and the alignment results may be sensitive to the choice of fiducial points and quality of training examples.

An alternative to this supervised approach is to take a set of poorly aligned images (*e.g.*, images drawn from approximately the same distribution as the inputs to the recognition system) and attempt to make the images more similar to each other, using some measure of joint similarity such as entropy. This framework of iteratively transforming images to reduce the entropy of the set is known as congealing [6], and was originally applied to specific types of images such as binary handwritten characters and magnetic resonance image volumes [7–9]. Congealing was extended to work on complex, real-world object classes such as faces and cars [10]. However, this required a careful selection of hand-crafted feature representation (SIFT [11]) and soft clustering, and does not achieve as large of an improvement in verification accuracy as supervised alignment (LFW-a).

In this work, we propose a novel combination of unsupervised alignment and unsupervised feature learning, specifically by incorporating deep learning [12–14] into the congealing framework. Through deep learning, we can obtain a feature representation tuned to the statistics of the specific object class we wish to align, and capture the data at multiple scales by using multiple layers of a deep learning architecture. Further, we incorporate a group sparsity constraint into the deep learning algorithm, leading to a topographic organization on the learned filters, and show that this leads to improved alignment results. We apply our method to unconstrained face images and show that, using the aligned images, we achieve a significantly higher face verification accuracy than obtained both using the original face images and using the images produced by prior work in unsupervised alignment [10]. In addition, the accuracy surpasses that achieved using supervised fiducial points based alignment [3], and matches the accuracy using the LFW-a images produced by commercial supervised alignment.

## 2   Related Work

We review relevant work in unsupervised joint alignment and deep learning.

### 2.1   Unsupervised Joint Alignment

Cox *et al.* presented a variation of congealing for unsupervised alignment, where the entropy similarity measure is replaced with a least-squares similarity measure [15, 16]. Liu *et al.* extended congealing by modifying the objective function to allow for simultaneous alignment and clustering [17]. Mattar *et al.* developed a transformed variant of Bayesian infinite models that can also simultaneously align and cluster complex data sets [18]. Zhu *et al.* developed a method for non-rigid alignment using a model parameterized by mesh vertex coordinates in a deformable Lucas-Kanade formulation [19]. However, this technique requires additional supervision in the form of object part (*e.g.*, eye) detectors specific to the data to be aligned.

In this work, we chose to extend the original congealing method, rather than other alignment frameworks, for several reasons. The algorithm uses entropy as a measure of similarity, rather than variance or least squares, thus allowing for the alignment of data with multiple modes. Unlike other joint alignment procedures [15], the main loop scales linearly with the number of images to be aligned, allowing for a greater number of images to be jointly aligned, smoothing the optimization landscape. Finally, congealing requires only *very weak supervision* in the form of poorly aligned images. However, our proposed extensions, using features obtained from deep learning, could also be applied to other alignment algorithms that have only been used with a pixel intensity representation, such as [15, 16, 19].

### 2.2   Deep Learning

A deep belief network (DBN) is a generative graphical model consisting of a layer of visible units and multiple layers of hidden units, where each layer encodes statistical dependencies in the units in

the layer below [12]. DBNs and related unsupervised learning algorithms such as auto-encoders [13] and sparse coding [20, 21] have been used to learn higher-level feature representations from unlabeled data, suitable for use in tasks such as classification. These methods have been successfully applied to computer vision tasks [22–26], as well as audio recognition [27], natural language processing [28], and information retrieval [29]. To the best of our knowledge, our proposed method is the first to apply deep learning to the alignment problem.

DBNs are generally trained using images drawn from the same distribution as the test images, which in our case corresponds to learning from faces in the LFW training set. In many machine learning problems, however, we are given only a limited amount of labeled data, which can cause an overfitting problem. Thus, we also examine the strategy of self-taught learning [30] (related to semi-supervised learning [31]). The idea of self-taught learning is to use a large amount of unlabeled data from a distribution different from the labeled data, and *transfer* low-level structures that can be shared between unlabeled and labeled data. For generic object categorization, Raina *et al*. [30] and Lee *et al*. [23] have shown successful applications of self-taught learning, using sparse coding and deep belief networks to learn feature representations from natural images. In this paper, we examine whether self-taught learning can be successful for alignment tasks.

In addition, we augment the training procedure of DBNs by adding a group sparsity regularization term, leading to a set of learned filters with a *linear* topographic organization. This idea is closely related to the Group Lasso for regression [32] and Topographic ICA [33], and has been applied to sparse coding with basis functions that form a generally two-dimensional topological map [34]. We extend this method to basis functions that are learned in a convolutional manner, and to higher-order features obtained from a multi-layer convolutional DBN.

## 3    Methodology

We begin with a review of the congealing framework. We then show how deep learning can be incorporated into this framework using convolutional DBNs, and how the learning algorithm can be modified through group sparsity regularization to improve congealing performance.

### 3.1    Congealing

We first define two terms used in congealing, the distribution field (DF) and the location stack. Let $\mathcal{X} = \{1, 2, \ldots, M\}$ be the set of all feature values. For example, letting the feature space be intensity values, $M = 2$ for binary images and $M = 256$ for 8-bit grayscale images. A distribution field is a distribution over $\mathcal{X}$ at each location in the image representation; *e.g.*, for binary images, a DF would be a distribution over $\{0, 1\}$ at each pixel in the image. One can view the DF as a generative independent pixel model of images, by placing a random variable $X_i$ at each pixel location $i$. An image then consists of a draw from the alphabet $\mathcal{X}$ for each $X_i$ according to the distribution over $\mathcal{X}$ at the $i$th pixel of the DF. Given a set of images, the location stack is defined as the set of values, with domain $\mathcal{X}$, at a specific location across a set of images. Thus, the empirical distribution at a given location of a DF is determined by the corresponding location stack.

Congealing proceeds by iteratively computing the empirical distribution defined by a set of images, then for each image, choosing a transformation (we use the set of similarity transformations) that reduces the entropy of the distribution field. Figure 2 illustrates congealing on one dimensional binary images. Under an independent pixel model and uniform distribution over transformations, minimizing the entropy of the distribution field is equivalent to maximizing the likelihood according to the distribution field [6].

Once congealing has been performed on a set of images (*e.g.*, a training set), *funneling* [6, 10] can be used to quickly align additional images, such as from a new test set. This is done by maintaining the

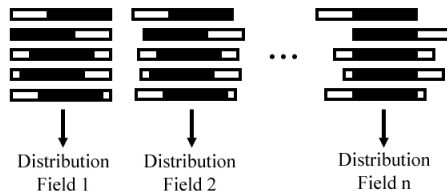

Figure 2: Schematic illustration of congealing of one dimensional binary images, where the transformation space is left-right translation

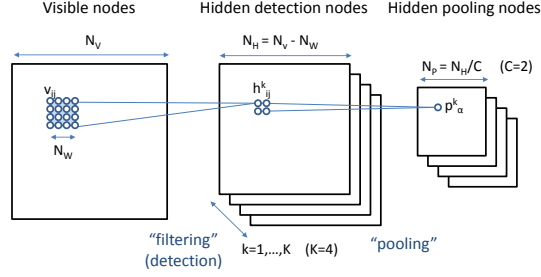

Figure 3: Illustration of convolutional RBM with probabilistic max-pooling. For illustration, we used pooling ratio $C = 2$ and number of filters $K = 4$. See text for details.

sequence of DFs from each iteration of congealing. A new image is then aligned by transforming it iteratively according to the sequence of saved DFs, thereby approximating the results of congealing on the original set of images as well as the new test image. As mentioned earlier, congealing was extended to work on complex object classes, such as faces, by using soft clustering of SIFT descriptors as the feature representation [10]. We will refer to this congealing algorithm as SIFT congealing. We now describe our proposed extension, which we refer to as *deep congealing*.

## 3.2   Deep Congealing

To incorporate deep learning within congealing, we use the convolutional restricted Boltzmann machine (CRBM) [23,35] and convolutional deep belief network (CDBN) [23]. The CRBM is an extension of the restricted Boltzmann machine, which is a Markov random field with a hidden layer and a visible layer (corresponding to image pixels in computer vision problems), where the connection between layers is bipartite. In the CRBM, rather than fully connecting the hidden layer and visible layer, the weights between the hidden units and the visible units are local (*i.e.*, $10 \times 10$ pixels instead of full image) and shared among all hidden units. An illustration of CRBM can be found in Figure 3. The CRBM has three sets of parameters: (1) $K$ convolution filter weights between the hidden nodes and the visible nodes, where each filter is $N_W \times N_W$ pixels (*i.e.*, $W^k \in \mathbb{R}^{N_W \times N_W}, k = 1, ..., K$); (2) hidden biases $b^k \in \mathbb{R}$ that are shared among hidden nodes; and (3) visible bias $c \in \mathbb{R}$ that is shared among visible nodes.

To make CRBMs more scalable, Lee *et al*. developed *probabilistic max-pooling*, a technique for incorporating local translation invariance. Max-pooling refers to operations where a local neighborhood (*e.g.*, $2 \times 2$ grid) of feature detection outputs is shrunk to a pooling node by computing the maximum of the local neighbors. Max-pooling makes the feature representation more invariant to local translations in the input data, and has been shown to be useful in computer vision [23,25,36].

Letting $P(\mathbf{v}, \mathbf{h}) = \frac{1}{Z} \exp(-E(\mathbf{v}, \mathbf{h}))$, we define the energy function of the probabilistic max-pooling CRBM as follows:[2]

$$E(\mathbf{v}, \mathbf{h}) = -\sum_{k=1}^{K} \sum_{i,j=1}^{N_H} h_{ij}^k (\tilde{W}^k * \mathbf{v})_{ij} + \sum_{r,s=1}^{N_V} \frac{1}{2} v_{rs}^2 - \sum_{k=1}^{K} b^k \sum_{i,j=1}^{N_H} h_{ij}^k - c \sum_{r,s=1}^{N_V} v_{rs}$$

$$\text{s.t.} \quad \sum_{(i,j) \in B_\alpha} h_{ij}^k \leq 1, \quad \forall k, \alpha$$

Here, $\tilde{W}^k$ refers to flipping the original filter $W^k$ in both upside-down and left-right directions, and $*$ denotes convolution. $B_\alpha$ refers to a $C \times C$ block of locally neighboring hidden units (*i.e.*, pooling region) $h_{i,j}^k$ that are pooled to a pooling node $p_\alpha^k$. The CRBM can be trained by approximately maximizing the log-likelihood of the unlabeled data via contrastive divergence [37]. For details on learning and inference in CRBMs, see [23].

After training a CRBM, we can use it to compute the posterior of the pooling units given the input data. These pooling unit activations can be used as input to further train the next layer CRBM. By stacking the CRBMs, the algorithm can capture high-level features, such as hierarchical object-part decompositions. After constructing a convolutional deep belief network, we perform (approximate)

inference of the whole network in a feedforward (bottom-up) manner. Specifically, letting $I(h_{ij}^k) \triangleq b^k + (\tilde{W}^k * \mathbf{v})_{ij}$, we can infer the pooling unit activations as a softmax function:

$$P(p_\alpha^k = 1|\mathbf{v}) = \frac{\sum_{(i',j') \in B_\alpha} \exp(I(h_{i'j'}^k))}{1 + \sum_{(i',j') \in B_\alpha} \exp(I(h_{i'j'}^k))}$$

Given a set of poorly aligned face images, our goal is to iteratively transform each image to reduce the total entropy over the pooling layer outputs of a CDBN applied to each of the images. For a CDBN with $K$ pooling layer groups, we now have $K$ location stacks at each image location (after max-pooling), over a binary distribution for each location stack. Given $N$ unaligned face images, let $P$ be the number of pooling units in each group in the top-most layer of the CDBN. We use the pooling unit probabilities, with the interpretation that the pooling unit can be considered as a mixture of sub-units that are on and off [6]. Letting $p_\alpha^{k,(n)}$ be the pooling unit $\alpha$ in group $k$ for image $n$ under some transformation $T^n$, we define $D_\alpha^k(1) = \frac{1}{N} \sum_{n=1}^N p_\alpha^{k,(n)}$ and $D_\alpha^k(0) = 1 - D_\alpha^k(1)$. Then, the entropy for a specific pooling unit is $H(D_\alpha^k) = -\sum_{s \in \{0,1\}} D_\alpha^k(s) \log(D_\alpha^k(s))$. At each iteration of congealing, we find a transformation for each image that decreases the total entropy $\sum_{k=1}^K \sum_{\alpha=1}^P H(D_\alpha^k)$. Note that if $K = 1$, this reduces to the traditional congealing formulation on the binary output of the single pooling layer.

### 3.3 Learning a Topology

As congealing reduces entropy by performing local hill-climbing in the transformation parameters, a key factor in the success of congealing is the smoothness of this optimization landscape. In SIFT congealing, smoothness is achieved through soft clustering and the properties of the SIFT descriptor. Specifically, to compute the descriptor, the gradient is computed at each pixel location and added to a weighted histogram over a fixed number of angles. The histogram bins have a natural circular topology. Therefore, the gradient at each location contributes to two neighboring histogram bins, weighted using linear interpolation. This leads to a smoother optimization landscape when congealing. For instance, if a face is rotated a fraction of the correct angle to put it into a good alignment, there will be a corresponding partial decrease in entropy due to this interpolated weighting.

In contrast, there is no topology on the filters produced using standard learning of a CRBM. This may lead to plateaus or local minima in the optimization landscape with congealing, for instance, if one filter is a small rotation of another filter, and a rotation of the image causes a section of the face to be between these two filters. This problem may be particularly severe for filters learned at deeper layers of a CDBN. For instance, a second-layer CDBN trained on face images would likely learn multiple filters that resemble eye detectors, capturing slightly different types and scales of eyes. If these filters are activating independently, then the resulting entropy of a set of images may not decrease even if eyes in different images are brought into closer alignment.

A CRBM is generally trained with sparsity regularization [38], such that each filter responds to a sparse set of input stimuli. A smooth optimization for congealing requires that, as an image patch is transformed from one such sparse set to another, the change in pooling unit activations is also gradual rather than abrupt. Therefore, we would like to learn filters with a linear topological ordering, such that when a particular pooling unit $p_\alpha^k$ at location $\alpha$ and associated with filter $k$ is activated, the pooling units at the same location, associated with nearby filters, *i.e.*, $p_\alpha^{k'}$ for $k'$ close to $k$, will also have partial activation. To learn a topology on the learned filters, we add the following group sparsity penalty to the learning objective function (*i.e.*, negative log-likelihood): $\mathcal{L}_{\text{sparsity}} = \lambda \sum_{k,\alpha} \sqrt{\sum_{k'} \omega_{k'-k}(p_\alpha^k)^2}$, where $\omega_d$ is a Gaussian weighting, $\omega_d \propto \exp(-\frac{d^2}{2\sigma^2})$. Let the term *array* be used to refer to the set of pooling units associated with a particular filter, *i.e.*, $p_\alpha^k$ for all locations $\alpha$. This regularization penalty is a sum ($L^1$ norm) of $L^2$ norms, each of which is a Gaussian weighting, centered at a particular array, of the pooling units across each array at a specific location. In practice, rather than weighting every array in each summand, we use a fixed kernel covering five consecutive filters, *i.e.*, $\omega_d = 0$ for $|d| > 2$.

The rationale behind such a regularization term is that, unlike an $L^2$ norm, an $L^1$ norm encourages sparsity. This sum of $L^2$ norms thus encourages sparsity at the group level, where a group is a set of Gaussian weighted activations centered at a particular array. Therefore, if two filters are similar and tend to both activate for the same visible data, a smaller penalty will be incurred if these filters

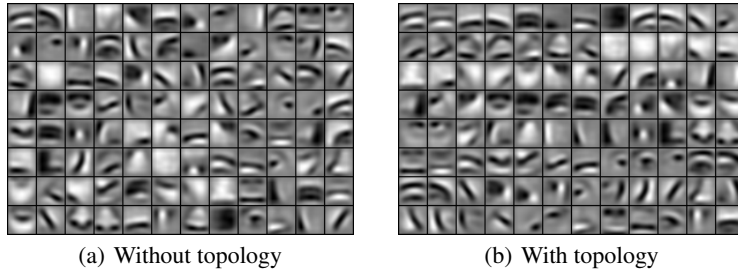

| (a) Without topology | (b) With topology |

Figure 4: Visualization of second layer filters learned from face images, without topology (left) and with topology (right). By learning with a linear topology, nearby filters (in row major order) have correlated activations. This leads to filters for particular facial features to be grouped together, such as eye detectors at the end of the row third from the bottom.

are nearby in the topological ordering, as this will lead to a more sparse representation at the group $L^2$ level. To account for this penalty term, we augment the learning algorithm by taking a step in the negative derivative with respect to the CRBM weights. We define $\alpha(i,j)$ as the pooling location associated with position $(i,j)$, and $J$ as $J_{ij}^{k,k'} = \frac{1}{\sqrt{\sum_{k''} \omega_{k''-k'}(p_{\alpha(i,j)}^{k''})^2}} p_{\alpha(i,j)}^k (1 - p_{\alpha(i,j)}^k) h_{ij}^k$. We can write the full gradient as $\nabla_{W^k} \mathcal{L}_{\text{sparsity}} = \lambda \sum_{k'} \omega_{k-k'}(v * \tilde{J}^{k,k'})$, where $*$ denotes convolution and $\tilde{J}^{k,k'}$ means $J^{k,k'}$ flipped horizontally and vertically. Thus we can efficiently compute the gradient as a sum of convolutions.

Following the procedure given by Sohn *et al.* [39], we initialize the filters using expectation-maximization under a mixture of Gaussians/Bernoullis, before proceeding with CRBM learning. Therefore, when learning with the group sparsity penalty, we periodically reorder the filters using the following greedy strategy. Taking the first filter, we iteratively add filters one by one to the end of the filter set, picking the filter that minimizes the group sparsity penalty.

## 4   Experiments

We learn three different convolutional DBN models to use as the feature representation for deep congealing. First, we learn a one-layer CRBM from the Kyoto images,[3] a standard natural image data set, to evaluate the performance of congealing with self-taught CRBM features. Next, we learn a one-layer CRBM from LFW face images, to compare performance when learning the features directly on images of the object class to be aligned. Finally, we learn a two-layer CDBN from LFW face images, to evaluate performance using higher-order features. For all three models, we also compare learning the weights using the standard sparse CDBN learning, as well as learning with group sparsity regularization. Visualizations of the top layer weights of the two-layer CDBN are given in Figure 4, demonstrating the effect of adding the sparsity regularization term.

We used $K = 32$ filters for the one-layer models and $K = 96$ in the top layer of the two-layer models. During learning, we used a pooling size of 5x5 for the one-layer models and 3x3 in both layers of the two-layer model. We used $\sigma^2 = 1$ in the Gaussian weighting for group sparsity regularization. For computing the pooling layer representation to use in congealing, we modified the pooling size to 3x3 for the one-layer models and 2x2 for the second layer in the two-layer model, and adjusted the hidden biases to give an expected activation of 0.025 for the hidden units. In Figure 5, we show a selection of images under several alignment methods. Each image is shown in its original form, and aligned using SIFT Congealing, Deep Congealing with topology, using a one-layer and two-layer CDBN trained on faces, and the LFW-a alignment.

We evaluate the effect of alignment on verification accuracy using View 1 of LFW. For the congealing methods, 400 images from the training set were congealed and used to form a funnel to subsequently align all of the images in both the training and test sets. To obtain verification accuracy, we use a variation on the method of Cosine Similarity Metric Learning (CSML) [40], one of the top-performing methods on LFW. As in CSML, we first apply whitening PCA and reduce the representation to 500 dimensions. We then normalize each image feature vector, and apply a linear SVM to an image pair by combining the image feature vectors using element-wise multiplication.

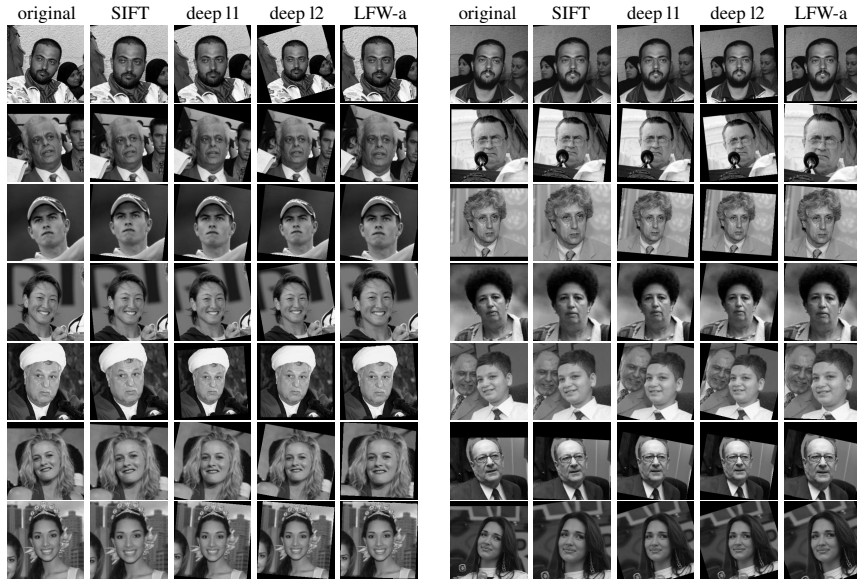

| original | SIFT | deep l1 | deep l2 | LFW-a | original | SIFT | deep l1 | deep l2 | LFW-a |

Figure 5: Sample images from LFW produced by different alignment algorithms. For each set of five images, the alignments are, from left to right: original images; SIFT Congealing; Deep Congealing, Faces, layer 1, with topology; Deep Congealing, Faces, layer 2, with topology; Supervised (LFW-a).

Note that if the weights of the SVM are 1 and the bias is 0, then this is equivalent to cosine similarity. We find that this procedure yields comparable accuracy to CSML but is much faster and less sensitive to the regularization parameters.[4] As our goal is to improve verification accuracy through better alignment, we focus on performance using a single feature representation, and only use the square root LBP features [40, 41] on 150x80 croppings of the full LFW images.

Table 1 gives the verification accuracy for this verification system using images produced by a number of alignment algorithms. Deep congealing gives a significant improvement over SIFT congealing. Using a CDBN representation learned with a group sparsity penalty, leading to learned filters with topographic organization, consistently gives a higher accuracy of one to two percentage points. We compare with two supervised alignment systems, the fiducial points based system of [3],[5] and LFW-a. Note that LFW-a was produced by a commercial alignment system, in the spirit of [3], but with important differences that have not been published [2]. Congealing with a one-layer CDBN[6] trained on faces, with topology, gives verification accuracy significantly higher than using images produced by [3], and comparable to the accuracy using LFW-a images.

Moreover, we can combine the verification scores using images from the one-layer and two-layer CDBN trained on faces, learning a second SVM on these scores. By doing so, we achieve a further gain in verification performance, achieving an accuracy of 0.831, exceeding the accuracy using LFW-a. This suggests that the two-layer CDBN alignment is somewhat complementary to the one-layer alignment. That is, although the two-layer CDBN alignment produces a lower verification accuracy, it is not strictly worse than the one-layer CDBN alignment for all images, but rather is aligning according to a different set of statistics, and achieves success on a different subset of images than the one-layer CDBN model. As a control, we performed the same score combination using the scores produced from images from the one-layer CDBN alignment trained on faces, with topology, and the original images. This gave a verification accuracy of 0.817, indicating that the improvement from combining two-layer scores is not merely obtained from using two different sets of alignments.

[4] We note that the accuracy published in [40] was higher than we were able to obtain in our own implementation. After communicating with the authors, we found that they used a different training procedure than described in the paper, which we believe inadvertently uses some test data as training, due to View 1 and View 2 of LFW not being mutually exclusive. Following the training procedure detailed in the paper, which we view to be correct, we find the accuracy to be about 3% lower than the published results.

[5] Using code available at http://www.robots.ox.ac.uk/~vgg/research/nface/

[6] Technically speaking, the term "one-layer CDBN" denotes a CRBM.

Table 1: Unconstrained face verification accuracy on View 1 of LFW using images produced by different alignment algorithms. By combining the classifier scores produced by layer 1 and 2 using a linear SVM, we achieve higher accuracy using unsupervised alignment than obtained using the widely-used LFW-a images, generated using a commercial supervised fiducial-points algorithm.

| Alignment | Accuracy |
|---|---|
| Original | 0.742 |
| SIFT Congealing | 0.758 |
| Deep Congealing, Kyoto, layer 1 | 0.807 |
| Deep Congealing, Kyoto, layer 1, with topology | 0.815 |
| Deep Congealing, Faces, layer 1 | 0.802 |
| Deep Congealing, Faces, layer 1, with topology | 0.820 |
| Deep Congealing, Faces, layer 2 | 0.780 |
| Deep Congealing, Faces, layer 2, with topology | 0.797 |
| Combining Scores of Faces, layers 1 and 2, with topology | **0.831** |
| Fiducial Points-based Alignment [3] (supervised) | 0.805 |
| LFW-a (commercial) | 0.823 |

## 5 Conclusion

We have shown how to combine unsupervised joint alignment with unsupervised feature learning. By congealing on the pooling layer representation of a CDBN, we are able to achieve significant gains in verification accuracy over existing methods for unsupervised alignment. By adding a group sparsity penalty to the CDBN learning algorithm, we can learn filters with a linear topology, providing a smoother optimization landscape for congealing. Using face images aligned by this method, we obtain higher verification accuracy than the supervised fiducial points based method of [3]. Further, despite being unsupervised, our method is still able to achieve comparable accuracy to the widely used LFW-a images, obtained by a commercial fiducial point-based alignment system whose detailed procedure is unpublished. We believe that our proposed method is an important contribution in developing generic alignment systems that do not require domain-specific fiducial points.

## Footnotes

[1] http://www.openu.ac.il/home/hassner/data/lfwa/

[2]We use real-valued visible units in the first-layer CRBM; however, we use binary-valued visible units when constructing the second-layer CRBM. See [23] for details.

[3]http://www.cnbc.cmu.edu/cplab/data_kyoto.html

## References

[1] L. Wolf, T. Hassner, and Y. Taigman. Similarity scores based on background samples. In *ACCV*, 2009.

[2] Y. Taigman, L. Wolf, and T. Hassner. Multiple one-shots for utilizing class label information. In *BMVC*, 2009.

[3] M. Everingham, J. Sivic, and A. Zisserman. "Hello! My name is... Buffy" - automatic naming of characters in TV video. In *BMVC*, 2006.

[4] T. L. Berg, A. C. Berg, M. Maire, R. White, Y. W. Teh, E. Learned-Miller, and D. A. Forsyth. Names and faces in the news. In *CVPR*, 2004.

[5] Y. Zhou, L. Gu, and H.-J. Zhang. Bayesian tangent shape model: Estimating shape and pose parameters via Bayesian inference. In *CVPR*, 2003.

[6] E. Learned-Miller. Data driven image models through continuous joint alignment. *PAMI*, 2005.

[7] E. Miller, N. Matsakis, and P. Viola. Learning from one example through shared densities on transforms. In *CVPR*, 2000.

[8] L. Zollei, E. Learned-Miller, E. Grimson, and W. Wells. Efficient population registration of 3d data. In *Workshop on Computer Vision for Biomedical Image Applications: Current Techniques and Future Trends, at ICCV*, 2005.

[9] E. Learned-Miller and V. Jain. Many heads are better than one: Jointly removing bias from multiple MRIs using nonparametric maximum likelihood. In *Proceedings of Information Processing in Medical Imaging*, pages 615–626, 2005.

[10] G. B. Huang, V. Jain, and E. Learned-Miller. Unsupervised joint alignment of complex images. In *ICCV*, 2007.

[11] D. G. Lowe. Distinctive image features from scale-invariant keypoints. *IJCV*, 60(2):91–110, 2004.

[12] G. E. Hinton, S. Osindero, and Y.-W. Teh. A fast learning algorithm for deep belief nets. *Neural Computation*, 18(7):1527–1554, 2006.

[13] Y. Bengio, P. Lamblin, D. Popovici, and H. Larochelle. Greedy layer-wise training of deep networks. In *NIPS*, 2007.

[14] M. Ranzato, Y.-L. Boureau, and Y. LeCun. Sparse feature learning for deep belief networks. In *NIPS*, 2007.

[15] M. Cox, S. Lucey, S. Sridharan, and J. Cohn. Least squares congealing for unsupervised alignment of images. In *CVPR*, 2008.

[16] M. Cox, S. Sridharan, S. Lucey, and J. Cohn. Least squares congealing for large numbers of images. In *ICCV*, 2009.

[17] X. Liu, Y. Tong, and F. W. Wheeler. Simultaneous alignment and clustering for an image ensemble. In *ICCV*, 2009.

[18] M. A. Mattar, A. R. Hanson, and E. G. Learned-Miller. Unsupervised joint alignment and clustering using Bayesian nonparametrics. In *UAI*, 2012.

[19] J. Zhu, L. V. Gool, and S. C. Hoi. Unsupervised face alignment by nonrigid mapping. In *ICCV*, 2009.

[20] B. A. Olshausen and D. J. Field. Emergence of simple-cell receptive field properties by learning a sparse code for natural images. *Nature*, 381:607–609, 1996.

[21] H. Lee, A. Battle, R. Raina, and A. Y. Ng. Efficient sparse coding algorithms. In *NIPS*, 2007.

[22] M. Zeiler, D. Krishnan, G. Taylor, and R. Fergus. Deconvolutional networks. In *CVPR*, 2010.

[23] H. Lee, R. Grosse, R. Ranganath, and A. Y. Ng. Unsupervised learning of hierarchical representations with convolutional deep belief networks. *Communications of the ACM*, 54(10):95–103, 2011.

[24] J. Yang, K. Yu, Y. Gong, and T. S. Huang. Linear spatial pyramid matching using sparse coding for image classification. In *CVPR*, pages 1794–1801, 2009.

[25] K. Jarrett, K. Kavukcuoglu, M. Ranzato, and Y. LeCun. What is the best multi-stage architecture for object recognition? In *ICCV*, 2009.

[26] G. B. Huang, H. Lee, and E. Learned-Miller. Learning hierarchical representations for face verification with convolutional deep belief networks. In *CVPR*, 2012.

[27] H. Lee, Y. Largman, P. Pham, and A. Y. Ng. Unsupervised feature learning for audio classification using convolutional deep belief networks. In *NIPS*, 2009.

[28] R. Collobert and J. Weston. A unified architecture for natural language processing: Deep neural networks with multitask learning. In *ICML*, 2008.

[29] R. Salakhutdinov and G. E. Hinton. Semantic hashing. *International Journal of Approximate Reasoning*, 50:969–978, 2009.

[30] R. Raina, A. Battle, H. Lee, B. Packer, and A. Y. Ng. Self-taught learning: Transfer learning from unlabeled data. In *ICML*, 2007.

[31] O. Chapelle, B. Schölkopf, and A. Zien. *Semi-supervised learning*. MIT Press, 2006.

[32] M. Yuan and L. Yin. Model selection and estimation in regression with grouped variables. Technical report, University of Wisconsin, 2004.

[33] A. Hyvärinen, P. O. Hoyer, and M. Inki. Topographic independent component analysis. *Neural Computation*, 13(7):1527–1558, 2001.

[34] K. Kavukcuoglu, M. Ranzato, R. Fergus, and Y. LeCun. Learning invariant features through topographic filter maps. In *CVPR*, 2009.

[35] M. Norouzi, M. Ranjbar, and G. Mori. Stacks of convolutional restricted Boltzmann machines for shift-invariant feature learning. In *CVPR*, pages 2735–2742, 2009.

[36] Y.-L. Boureau, F. R. Bach, Y. LeCun, and J. Ponce. Learning mid-level features for recognition. In *CVPR*, 2010.

[37] G. E. Hinton. Training products of experts by minimizing contrastive divergence. *Neural Computation*, 14(8):1771–1800, 2002.

[38] H. Lee, C. Ekanadham, and A. Y. Ng. Sparse deep belief net model for visual area V2. In *NIPS*, 2008.

[39] K. Sohn, D. Y. Jung, H. Lee, and A. H. III. Efficient learning of sparse, distributed, convolutional feature representations for object recognition. In *ICCV*, 2011.

[40] H. V. Nguyen and L. Bai. Cosine similarity metric learning for face verification. In *ACCV*, 2010.

[41] T. Ojala, M. Pietikinen, and D. Harwood. A comparative study of texture measures with classification based on feature distributions. *Pattern Recognition*, 19(3):51–59, 1996.

